# Continuous sigmoidal belief networks trained using slice sampling

**Brendan J. Frey**
Department of Computer Science, University of Toronto
6 King's College Road, Toronto, Canada M5S 1A4

## Abstract

Real-valued random hidden variables can be useful for modelling latent structure that explains correlations among observed variables. I propose a simple unit that adds zero-mean Gaussian noise to its input before passing it through a sigmoidal squashing function. Such units can produce a variety of useful behaviors, ranging from deterministic to binary stochastic to continuous stochastic. I show how "slice sampling" can be used for inference and learning in top-down networks of these units and demonstrate learning on two simple problems.

## 1 Introduction

A variety of unsupervised connectionist models containing discrete-valued hidden units have been developed. These include Boltzmann machines (Hinton and Sejnowski 1986), binary sigmoidal belief networks (Neal 1992) and Helmholtz machines (Hinton *et al.* 1995; Dayan *et al.* 1995). However, some hidden variables, such as translation or scaling in images of shapes, are best represented using continuous values. Continuous-valued Boltzmann machines have been developed (Movellan and McClelland 1993), but these suffer from long simulation settling times and the requirement of a "negative phase" during learning. Tibshirani (1992) and Bishop *et al.* (1996) consider learning mappings from a *continuous* latent variable space to a higher-dimensional input space. MacKay (1995) has developed "density networks" that can model both continuous and categorical latent spaces using stochasticity at the top-most network layer. In this paper I consider a new hierarchical top-down connectionist model that has stochastic hidden variables at all layers; moreover, these variables can *adapt* to be continuous or categorical.

The proposed top-down model can be viewed as a continuous-valued belief network, which can be simulated by performing a quick top-down pass (Pearl 1988). Work done on continuous-valued belief networks has focussed mainly on Gaussian random variables that are linked linearly such that the joint distribution over all

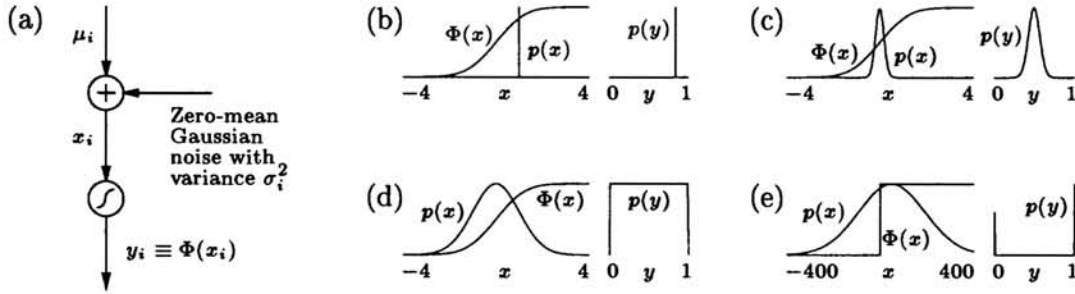

Figure 1: (a) shows the inner workings of the proposed unit. (b) to (e) illustrate four quite different modes of behavior: (b) deterministic mode; (c) stochastic linear mode; (d) stochastic nonlinear mode; and (e) stochastic binary mode (note the different horizontal scale). For the sake of graphical clarity, the density functions are normalized to have equal maxima and the subscripts are left off the variables.

variables is also Gaussian (Pearl 1988; Heckerman and Geiger 1995). Lauritzen *et al.* (1990) have included discrete random variables within the linear Gaussian framework. These approaches infer the distribution over unobserved unit activities given observed ones by "probability propagation" (Pearl 1988). However, this procedure is highly suboptimal for the richly connected networks that I am interested in. Also, these approaches tend to assume that all the conditional Gaussian distributions represented by the belief network can be easily derived using information elicited from experts. Hofmann and Tresp (1996) consider the case of inference and learning in continuous belief networks that may be richly connected. They use mixture models and Parzen windows to implement conditional densities.

My main contribution is a simple, but versatile, continuous random unit that can operate in several different modes ranging from deterministic to binary stochastic to continuous stochastic. This spectrum of behaviors is controlled by only two parameters. Whereas the above approaches assume a particular mode for each unit (Gaussian or discrete), the proposed units are capable of adapting in order to operate in whatever mode is most appropriate.

## 2   Description of the unit

The proposed unit is shown in Figure 1a. It is similar to the deterministic sigmoidal unit used in multilayer perceptrons, except that Gaussian noise is added to the total input, $\mu_i$, before the sigmoidal squashing function is applied.[1] The probability density over *presigmoid* activity $x_i$ for unit $i$ is

$$p(x_i|\mu_i, \sigma_i^2) \equiv \exp[-(x_i - \mu_i)^2/2\sigma_i^2]/\sqrt{2\pi\sigma_i^2}, \tag{1}$$

where $\mu_i$ and $\sigma_i^2$ are the mean and variance for unit $i$. A *postsigmoid* activity, $y_i$, is obtained by passing the presigmoid activity through a sigmoidal squashing function:

$$y_i \equiv \Phi(x_i). \tag{2}$$

Including the transformation Jacobian, the postsigmoid distribution for unit $i$ is

$$p(y_i|\mu_i, \sigma_i^2) = \frac{\exp[-(\Phi^{-1}(y_i) - \mu_i)^2/2\sigma_i^2]}{\Phi'(\Phi^{-1}(y_i))\sqrt{2\pi\sigma_i^2}}. \tag{3}$$

I use the cumulative Gaussian squashing function:

$$\Phi(x) \equiv \int_{-\infty}^{x} e^{-z^2/2}/\sqrt{2\pi}\, dz \qquad \Phi'(x) = \phi(x) \equiv e^{-x^2/2}/\sqrt{2\pi}. \qquad (4)$$

Both $\Phi()$ and $\Phi^{-1}()$ are nonanalytic, so I use the C-library erf() function to implement $\Phi()$ and table lookup with quadratic interpolation to implement $\Phi^{-1}()$.

Networks of these units can *represent* a broad range of structures, including deterministic multilayer perceptrons, binary sigmoidal belief networks (*aka*. stochastic multilayer perceptrons), mixture models, mixture of expert models, hierarchical mixture of expert models, and factor analysis models. This versatility is brought about by a range of significantly different modes of behavior available to each unit. Figures 1b to 1e illustrate these modes.

**Deterministic mode:** If the noise variance of a unit is very small, the postsigmoid activity will be a practically deterministic sigmoidal function of the mean. This mode is useful for representing deterministic nonlinear mappings such as those found in deterministic multilayer perceptrons and mixture of expert models.

**Stochastic linear mode:** For a given mean, if the squashing function is approximately linear over the span of the added noise, the postsigmoid distribution will be approximately Gaussian with the mean and standard deviation linearly transformed. This mode is useful for representing Gaussian noise effects such as those found in mixture models, the outputs of mixture of expert models, and factor analysis models.

**Stochastic nonlinear mode:** If the variance of a unit in the stochastic linear mode is increased so that the squashing function is used in its nonlinear region, a variety of distributions are producible that range from skewed Gaussian to uniform to bimodal.

**Stochastic binary mode:** This is an extreme case of the stochastic nonlinear mode. If the variance of a unit is very large, then nearly all of the probability mass will lie near the ends of the interval $(0, 1)$ (see figure 1e). Using the cumulative Gaussian squashing function and a standard deviation of 150, less than 1% of the mass lies in $(0.1, 0.9)$. In this mode, the postsigmoid activity of unit $i$ appears to be binary with probability of being "on" (*ie.*, $y_i > 0.5$ or, equivalently, $x_i > 0$):

$$p(i \text{ on}|\mu_i, \sigma_i^2) = \int_0^\infty \frac{\exp[-(x-\mu_i)^2/2\sigma_i^2]}{\sqrt{2\pi\sigma_i^2}}\, dx = \int_{-\infty}^{\mu_i} \frac{\exp[-x^2/2\sigma_i^2]}{\sqrt{2\pi\sigma_i^2}}\, dx = \Phi\left(\frac{\mu_i}{\sigma_i}\right). \quad (5)$$

This sort of stochastic activation is found in binary sigmoidal belief networks (Jaakkola *et al.* 1996) and in the decision-making components of mixture of expert models and hierarchical mixture of expert models.

## 3   Continuous sigmoidal belief networks

If the mean of each unit depends on the activities of other units and there are feedback connections, it is difficult to relate the density in equation 3 to a joint distribution over all unit activities, and simulating the model would require a great deal of computational effort. However, when a top-down topology is imposed on the network (making it a directed acyclic graph), the densities given in equations 1 and 3 can be interpreted as conditional distributions and the joint distribution over all units can be expressed as

$$p(\{x_i\}) = \prod_{i=1}^{N} p(x_i|\{x_j\}_{j<i}) \quad \text{or} \quad p(\{y_i\}) = \prod_{i=1}^{N} p(y_i|\{y_j\}_{j<i}), \qquad (6)$$

where $N$ is the number of units. $p(x_i|\{x_j\}_{j<i})$ and $p(y_i|\{y_j\}_{j<i})$ are the presigmoid and postsigmoid densities of unit $i$ conditioned on the activities of units with lower

indices. This ordered arrangement is the foundation of belief networks (Pearl, 1988). I let the mean of each unit be determined by a linear combination of the postsigmoid activities of preceding units:

$$\mu_i = \sum_{j<i} w_{ij} y_j, \tag{7}$$

where $y_0 \equiv 1$ is used to implement biases. The variance for each unit is independent of unit activities. A single sample from the joint distribution can be obtained by using the bias as the mean for unit 1, randomly picking a noise value for unit 1, applying the squashing function, computing the mean for unit 2, picking a noise value for unit 2, and so on in a simple top-down pass.

## Inference by slice sampling

Given the activities of a set of visible (observed) units, $V$, inferring the distribution over the remaining set of hidden (unobserved) units, $H$, is in general a difficult task. The brute force procedure proceeds by obtaining the posterior density using Bayes theorem:

$$p(\{y_i\}_{i\in H} | \{y_i\}_{i\in V}) = p(\{y_i\}_{i\in H}, \{y_i\}_{i\in V}) / \int_{\{y_i\}_{i\in H}} p(\{y_i\}_{i\in H}, \{y_i\}_{i\in V}) \prod_{i\in H} dy_i. \tag{8}$$

However, computing the integral in the denominator exactly is computationally intractable for any more than a few hidden units. The combinatorial explosion encountered in the corresponding sum for discrete-valued belief networks pales in comparison to this integral; not only is it combinatorial, but it is a continuous integral with a multimodal integrand whose peaks may be broad in some dimensions but narrow in others, depending on what modes the units are in.

An alternative to explicit integration is to sample from the posterior distribution using Markov chain Monte Carlo. Given a set of observed activities, this procedure produces a state sequence, $\{y_i\}_{i\in H}^{(0)}, \{y_i\}_{i\in H}^{(1)}, ..., \{y_i\}_{i\in H}^{(t)}, ...,$ that is guaranteed to converge to the posterior distribution. Each successive state is randomly selected based on knowledge of only the previous state. To simplify these random choices, I consider changing only one unit at a time when making a state transition. Ideally, the new activity of unit $i$ would be drawn from the conditional distribution $p(y_i|\{y_j\}_{j\neq i})$ (Gibbs sampling). However, it is difficult to sample from this distribution because it may have many peaks that range from broad to narrow.

I use a new Markov chain Monte Carlo method called "slice sampling" (Neal 1997) to pick a new activity for each unit. Consider the problem of drawing a value $y$ from a univariate distribution $P(y)$ — in this application, $P(y)$ is the conditional distribution $p(y_i|\{y_j\}_{j\neq i})$. Slice sampling does not directly produce values distributed according to $P(y)$, but instead produces a sequence of values that is guaranteed to converge to $P(y)$. At each step in the sequence, the old value $y_{\text{old}}$ is used as a guide for where to pick the new value $y_{\text{new}}$.

To perform slice sampling, all that is needed is an efficient way to evaluate a function $f(y)$ that is *proportional* to $P(y)$ — in this application, the easily computed value $p(y_i, \{y_j\}_{j\neq i})$ suffices, since $p(y_i, \{y_j\}_{j\neq i}) \propto p(y_i|\{y_j\}_{j\neq i})$. Figure 2a shows an example of a univariate distribution, $P(y)$. The version of slice sampling that I use requires that all of the density lies within a bounded *interval* as shown. To obtain $y_{\text{new}}$ from $y_{\text{old}}$, $f(y_{\text{old}})$ is first computed and then a uniform random value is drawn from $[0, f(y_{\text{old}})]$. The distribution is then horizontally "sliced" at this value, as shown in figure 2a. Any $y$ for which $f(y)$ is greater than this value is considered to be part of the slice, as indicated by the bold line segments in the picture shown at the top of figure 2b. Ideally, $y_{\text{new}}$ would now be drawn uniformly from the slice. However, determining the line segments that comprise the slice is not easy, for although it is easy to determine whether a particular $y$ is in the slice,

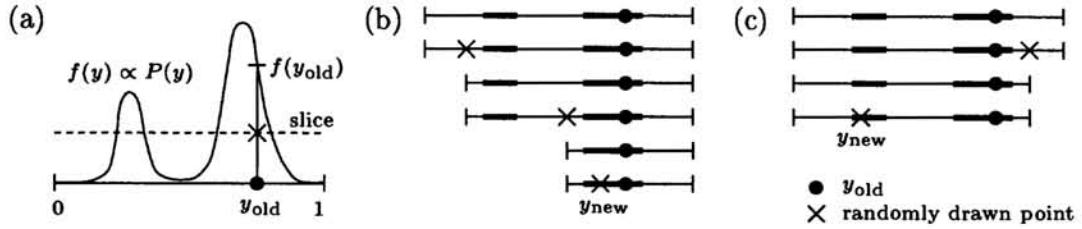

Figure 2: After obtaining a random slice from the density (a), random values are drawn until one is accepted. (b) and (c) show two such sequences.

it is much more difficult to determine the line segment boundaries, especially if the distribution is multimodal. Instead, a uniform value is drawn from the original interval as shown in the second picture of figure 2b. If this value is in the slice it is accepted as $y_{new}$ (note that this decision requires an evaluation of $f(y)$). Otherwise either the left or the right interval boundary is moved to this new value, while keeping $y_{old}$ in the interval. This procedure is repeated until a value is accepted. For the sequence in figure 2b, the new value is in the same mode as the old one, whereas for the sequence in figure 2c, the new value is in the other mode. Once $y_{new}$ is obtained, it is used as $y_{old}$ for the next step.

If the top-down influence causes there to be two very narrow peaks in $p(y_i|\{y_j\}_{j\neq i})$ (corresponding to a unit in the stochastic binary mode) the slices will almost always consist of two very short line segments and it will be very difficult for the above procedure to switch from one mode to another. To fix this problem, slice sampling is performed in a new domain, $z_i = \Phi(\{x_i - \mu_i\}/\sigma_i)$. In this domain the top-down distribution $p(z_i|\{y_j\}_{j<i})$ is uniform on $(0,1)$, so $p(z_i|\{y_j\}_{j\neq i}) = p(z_i|\{y_j\}_{j>i})$ and I use the following function for slice sampling:

$$f(z_i) = \exp\left[-\sum_{k=i+1}^{N}\left\{x_k - \mu_k^{-i} - w_{ki}\Phi\left(\sigma_i\Phi^{-1}(z_i) + \mu_i\right)\right\}^2/2\sigma_k^2\right], \qquad (9)$$

where $\mu_k^{-i} = \sum_{j<k, j\neq i} w_{kj}y_j$. Since $x_i$, $y_i$ and $z_i$ are all deterministically related, sampling from the distribution of $z_i$ will give appropriately distributed values for the other two. Many slice sampling steps could be performed to obtain a reliable sample from the conditional distribution for unit $i$, before moving on to the next unit. Instead, only one slice sampling step is performed for unit $i$ before moving on. The latter procedure often converges to the correct distribution more quickly than the former. The Markov chain Monte Carlo procedure I use in the simulations below thus consists of sweeping a prespecified number of times through the set of hidden units, while updating each unit using slice sampling.

## Learning

Given training examples indexed by $\tau$, I use on-line stochastic gradient ascent to perform maximum likelihood learning — ie., maximize $\prod_\tau p(\{x_i^\tau\}_{i\in V})$. This consists of sweeping through the training set and for each case $\tau$ following the gradient of $\ln p(\{x_i\})$, while sampling hidden unit values from $p(\{x_i\}_{i\in H}|\{x_i^\tau\}_{i\in V})$ using the sampling algorithm described above. From equations 1, 6 and 7,

$$\Delta w_{jk} \equiv \eta\, \partial\ln p(\{x_i\})/\partial w_{jk} = \eta\left(x_j - \sum_{l<j}w_{jl}y_l\right)y_k/\sigma_j^2, \qquad (10)$$

$$\Delta\ln\sigma_j^2 \equiv \eta\, \partial\ln p(\{x_i\})/\partial\ln\sigma_j^2 = \eta\left[\left(x_j - \sum_{l<j}w_{jl}y_l\right)^2/\sigma_j^2 - 1\right]/2, \qquad (11)$$

where $\eta$ is the learning rate.

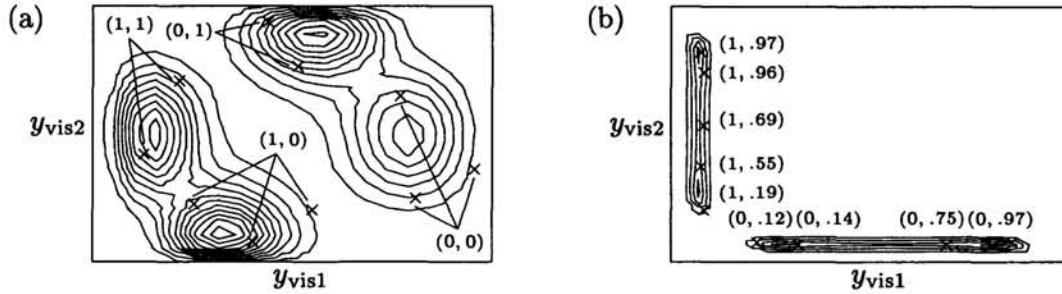

Figure 3: For each experiment (a) and (b), contours show the distribution of the 2-dimensional training cases. The inferred postsigmoid activities of the two hidden units after learning are shown in braces for several training cases, marked by ×.

## 4   Experiments

I designed two experiments meant to elicit the four modes of operation described in section 2. Both experiments were based on a simple network with one hidden layer containing two units and one visible layer containing two units. Training data was obtained by carefully selecting model parameters so as to induce various modes of operation and then generating 10,000 two-dimensional examples. Before training, the weights and biases were initialized to uniformly random values between -0.1 and 0.1; log-variances were initialized to 10.0. Training consisted of 100 epochs using a learning rate of 0.001 and 20 sweeps of slice sampling to complete each training case. Each task required roughly five minutes on a 200 MHz MIPS R4400 processor.

The distribution of the training cases in visible unit space $(y_{vis1} - y_{vis2})$ for the first experiment is shown by the contours in figure 3a. After training the network, I ran the inference algorithm for each of ten representative training cases. The postsigmoid activities of the two hidden units are shown beside the cases in figure 3a; clearly, the network has identified four classes that it labels $(0, 0)...(1, 1)$. Based on a 30x30 histogram, the Kullback-Leibler divergence between the training set and data generated from the trained network is 0.02 bits. Figure 3b shows a similar picture for the second experiment, using different training data. In this case, the network has identified two categories that it labels using the first postsigmoid activity. The second postsigmoid activity indicates how far along the respective "ridge" the data point lies. The Kullback-Leibler divergence in this case is 0.04 bits.

## 5   Discussion

The proposed continuous-valued nonlinear random unit is meant to be a useful atomic element for continuous belief networks in much the same way as the sigmoidal deterministic unit is a useful atomic element for multi-layer perceptrons. Four operational modes available to each unit allows small networks of these units to exhibit complex stochastic behaviors. The new "slice sampling" method that I employ for inference and learning in these networks uses easily computed local information.

The above experiments illustrate how the same network can be used to model two quite different types of data. In contrast, a Gaussian mixture model would require many more components for the second task as compared to the first. Although the methods due to Tibshirani and Bishop *et al.* would nicely model each submanifold in the second task, they would not properly distinguish between categories of data in either task. MacKay's method may be capable of extracting both the submanifolds and the categories, but I am not aware of any results on such a dual problem.

It is not difficult to conceive of models for which naive Markov chain Monte Carlo procedures will become fruitlessly slow. In particular, if two units have very highly correlated activities, the procedure of changing one activity at a time will converge extremely slowly. Also, the Markov chain method may be prohibitive for larger networks. One approach to avoiding these problems is to use the Helmholtz machine (Hinton *et al.* 1995) or mean field methods (Jaakkola *et al.* 1996).

Other variations on the theme presented in this paper include the use of other types of distributions for the hidden units (*e.g.*, Poisson variables may be more biologically plausible) and different ways of parameterizing the modes of behavior.

## Acknowledgements

I thank Radford Neal and Geoffrey Hinton for several essential suggestions and I also thank Peter Dayan and Tommi Jaakkola for helpful discussions. This research was supported by grants from ITRC, IRIS, and NSERC.

## Footnotes

[1]Geoffrey Hinton suggested this unit as a way to make factor analysis nonlinear.

## References

Bishop, C. M, Svensen, M., and Williams, C.K.I. 1996. EM optimization of latent-variable density models. In D. Touretzky, M. Mozer, and M. Hasselmo (editors), *Advances in Neural Information Processing Systems 8*, MIT Press, Cambridge, MA.

Dayan, P., Hinton, G. E., Neal, R. M., and Zemel, R. S. 1995. The Helmholtz machine. *Neural Computation* **7**, 889-904.

Heckerman, D., and Geiger, D. 1994. Learning Bayesian networks: a unification for discrete and Gaussian domains. In P. Besnard and S. Hanks (editors), *Proceedings of the Eleventh Conference on Uncertainty in Artificial Intelligence*, Morgan Kaufmann, San Francisco, CA, 274-284.

Hinton, G. E., Dayan, P., Frey, B. J., and Neal, R. M. 1995. The wake-sleep algorithm for unsupervised neural networks. *Science* **268**, 1158-1161.

Hinton, G. E., and Sejnowski, T. J. 1986. Learning and relearning in Boltzmann machines. In D. E. Rumelhart and J. L. McClelland (editors), *Parallel Distributed Processing: Explorations in the Microstructure of Cognition. Volume 1: Foundations*. MIT Press, Cambridge, MA.

Hofmann, R., and Tresp, V. 1996. Discovering structure in continuous variables using Bayesian networks. In D. Touretzky, M. Mozer, and M. Hasselmo (editors), *Advances in Neural Information Processing Systems 8*, MIT Press, Cambridge, MA.

Jaakkola, T., Saul, L. K., and Jordan, M. I. 1996. Fast learning by bounding likelihoods in sigmoid type belief networks. In D. Touretzky, M. Mozer and M. Hasselmo (editors), *Advances in Neural Information Processing Systems 8*, MIT Press, Cambridge, MA.

Lauritzen, S. L., Dawid, A. P., Larsen, B. N., and Leimer, H. G. 1990. Independence properties of directed Markov Fields. *Networks* **20**, 491-505.

MacKay, D. J. C. 1995. Bayesian neural networks and density networks. *Nuclear Instruments and Methods in Physics Research, A* **354**, 73-80.

Movellan, J. R., and McClelland, J. L. 1992. Learning continuous probability distributions with symmetric diffusion networks. *Cognitive Science* **17**, 463-496.

Neal, R. M. 1992. Connectionist learning of belief networks. *Artificial Intelligence* **56**, 71-113.

Neal, R. M. 1997. Markov chain Monte Carlo methods based on "slicing" the density function. In preparation.

Pearl, J. 1988. *Probabilistic Reasoning in Intelligent Systems: Networks of Plausible Inference*. Morgan Kaufmann, San Mateo, CA.

Tibshirani, R. (1992). Principal curves revisited. *Statistics and Computing* **2**, 183-190.